# A Micropower Analog VLSI HMM State Decoder for Wordspotting

**John Lazzaro and John Wawrzynek**
CS Division, UC Berkeley
Berkeley, CA 94720-1776
lazzaro@cs.berkeley.edu, johnw@cs.berkeley.edu

**Richard Lippmann**
MIT Lincoln Laboratory
Room S4–121, 244 Wood Street
Lexington, MA 02173-0073
rpl@sst.ll.mit.edu

## Abstract

We describe the implementation of a hidden Markov model state decoding system, a component for a wordspotting speech recognition system. The key specification for this state decoder design is microwatt power dissipation; this requirement led to a continuous-time, analog circuit implementation. We characterize the operation of a 10-word (81 state) state decoder test chip.

## 1. INTRODUCTION

In this paper, we describe an analog implementation of a common signal processing block in pattern recognition systems: a hidden Markov model (HMM) state decoder. The design is intended for applications such as voice interfaces for portable devices that require micropower operation. In this section, we review HMM state decoding in speech recognition systems.

An HMM speech recognition system consists of a probabilistic state machine, and a method for tracing the state transitions of the machine for an input speech waveform. Figure 1 shows a state machine for a simple recognition problem: detecting the presence of keywords ("Yes," "No") in conversational speech (non-keyword speech is captured by the "Filler" state). This type of recognition where keywords are detected in unconstrained speech is called wordspotting (Lippmann *et al.*, 1994).

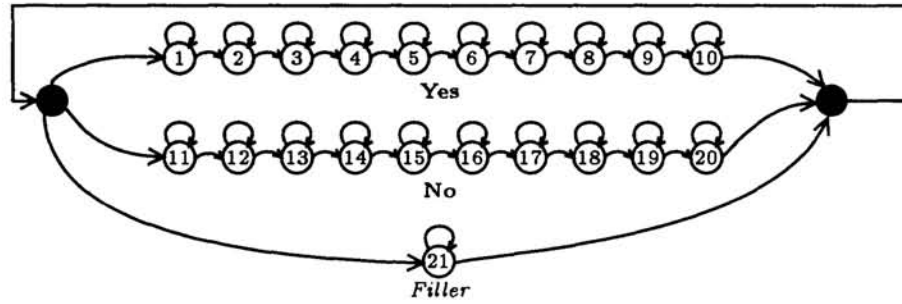

**Figure 1.** A two-keyword ("Yes," states 1-10, "No," states 11-20) HMM.

Our goal during speech recognition is to trace out the most likely path through this state machine that could have produced the input speech waveform. This problem can be partially solved in a local fashion, by examining short (80 ms. window) overlapping (15 ms. frame spacing) segments of the speech waveform. We estimate the probability $b_i(n)$ that the signal in frame $n$ was produced by state $i$, using static pattern recognition techniques.

To improve the accuracy of these local estimates, we need to integrate information over the entire word. We do this by creating a set of state variables for the machine, called likelihoods, that are incrementally updated at every frame. Each state $i$ has a real-valued likelihood $\phi_i(n)$ associated with it. Most states in Figure 1 have a stereotypical form: a state $i$ that has a self-loop input, an input from state $i-1$, and an output to state $i+1$, with the self-loop and exit transitions being equally probable. For states in this topology, the update rule

$$\log(\phi_i(n)) = \log(b_i(n)) + \log(\phi_i(n-1) + \phi_{i-1}(n-1)) \qquad (1)$$

lets us estimate the "log likelihood" value $\log(\phi_i(n))$ for the state $i$; a log encoding is used to cope with the large operating range of $\phi_i(n)$ values. Log likelihoods are negative numbers, whose magnitudes increase with each frame. We limit the range of log likelihood values by using a renormalization technique: if any log likelihood in the system falls below a minimum value, a positive constant is added to all log likelihoods in the machine.

Figure 2 shows a complete system which uses HMM state decoding to perform wordspotting. The "Feature Generation" and "Probability Generation" blocks comprise the static pattern recognition system, producing the probabilities $b_i(n)$ at each frame. The "State Decode" block updates the log likelihood variables $\log(\phi_i(n))$. The "Word Detect" block uses a simple online algorithm to flag the occurrence of a word. Keyword end-state log likelihoods are subtracted by the filler log likelihood, and when this difference exceeds a fixed threshold a keyword is detected.

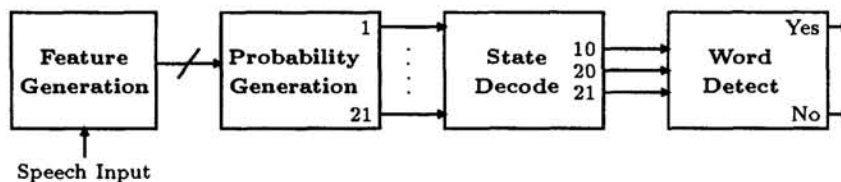

**Figure 2.** Block diagram for the two-keyword spotting system.

## 2. ANALOG CIRCUITS FOR STATE DECODING

Figure 3a shows an analog discrete-time implementation of Equation 1. The delay element (labeled $Z^{-1}$) acts as a edge-triggered sampled analog delay, with full-scale voltage input and output. The delay element is clocked at the frame rate of the state decoder (15 ms. clock period). The "combinatorial" analog circuits must settle within the clock period. A clock period of 15 ms. allows a relatively long settling time, which enables us to make extensive use of submicroampere currents in the circuit design. The microwatt power consumption design specification drives us to use such small currents. As a result of submicroampere circuit operation, the MOS transistors in Figure 3a are operating in the weak-inversion regime.

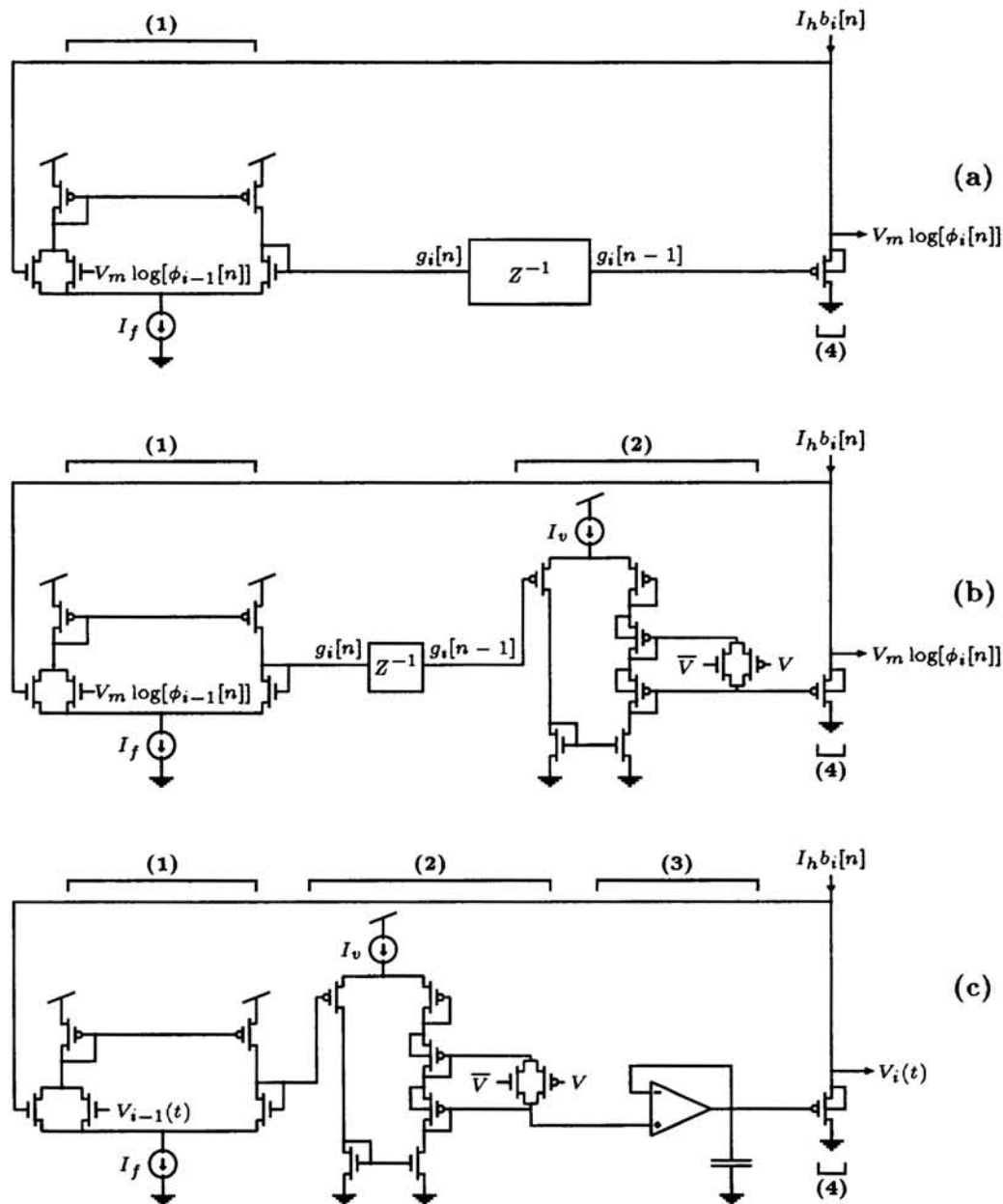

**Figure 3.** (a) Analog discrete-time single-state decoder. (b) Enhanced version of (a), includes the renormalization system. (c) Continuous-time extension of (b).

Equation 1 uses two types of variables: probabilities and log likelihoods. In the implementation shown in Figure 3, we choose unidirectional current as the signal type for probability, and large-signal voltage as the signal type for log likelihood. We can understand the dimensional scaling of these signal types by analyzing the floating-well transistor labeled (**4**) in Figure 3a. The equation

$$V_m \log(\phi_i(n)) = V_m \log(b_i(n)) + g_i(n-1) + V_m \log(\frac{I_h}{I_o}) \qquad (2)$$

describes the behavior of this transistor, where $V_m = (V_o/\kappa_p)\ln(10)$, $g_i(n-1)$ is the output of the delay element, and $I_o, \kappa$ and $V_o$ are MOS parameters. Both $I_o$ and $\kappa$ in Equation 2 are functions of $V_{sb}$. However, the floating-well topology of the transistor ensures $V_{sb} = 0$ for this device.

The input probability $b_i(n)$ is scaled by the unidirectional current $I_h$, defining the current flowing through the transistor. The current $I_h$ is the largest current that keeps the transistor in the weak-inversion regime. We define $I_l$ to be the smallest value for $I_h b_i(n)$ that allows the circuit to settle within the clock period. The ratio $I_h/I_l$ sets the supported range of $b_i(n)$. In the test-chip fabrication process, $I_h/I_l \approx 10,000$ is feasible, which is sufficient for accurate wordspotting. Likewise, the unitless $\log(\phi_i(n))$ is scaled by the voltage $V_m$ to form a large-signal voltage encoding of log likelihood. A nominal value for $V_m$ is 85mV in the test-chip process. To support a log likelihood range of 35 (the necessary range for accurate wordspotting) a large-signal voltage range of 3 volts (i.e. $35V_m$) is required.

The term $g_i(n-1)$ in Equation 2 is shown as the output of the circuit labeled (**1**) in Figure 3a. This circuit computes a function that approximates the desired expression $V_m\log(\phi_i(n-1) + \phi_{i-1}(n-1))$, if the transistors in the circuit operate in the weak-inversion regime.

The computed log likelihood $\log(\phi_i(n))$ in Equation 1 decreases every frame. The circuit shown in Figure 3a does not behave in this way: the voltage $V_m\log(\phi_i(n))$ *increases* every frame. This difference in behavior is attributable to the constant term $V_m\log(I_h/I_o)$ in Equation 2, which is not present in Equation 1, and is always larger than the negative contribution from $V_m\log(b_i(n))$. Figure 3b adds a new circuit (labeled (**2**)) to Figure 3a, that allows the constant term in Equation 2 to be altered under control of the binary input $V$. If $V$ is $V_{dd}$, the circuit in Figure 3b is described by

$$V_m \log(\phi_i(n)) = V_m \log(b_i(n)) + g_i(n-1) + V_m \log(\frac{I_h I_o}{I_v^2}), \qquad (3a)$$

where the term $V_m\log((I_h I_o)/I_v^2)$ should be less than or equal to zero. If $V$ is grounded, the circuit is described by

$$V_m \log(\phi_i(n)) = V_m \log(b_i(n)) + g_i(n-1) + V_m \log(\frac{I_h}{I_v}), \qquad (3b)$$

where the term $V_m\log(I_h/I_v)$ should have a positive value of at least several hundred millivolts. The goal of this design is to create two different operational modes for the system. One mode, described by Equation 3a, corresponds to the normal state decoder operation described in Equation 1. The other mode, described by Equation

3b, corresponds to the renormalization procedure, where a positive constant is added to all likelihoods in the system. During operation, a control system alternates between these two modes, to manage the dynamic range of the system.

Section 1 formulated HMMs as discrete-time systems. However, there are significant advantages in replacing the $Z^{-1}$ element in Figure 3b with a continuous-time delay circuit. The switching noise of a sampled delay is eliminated. The power consumption and cell area specifications also benefit from continuous-time implementation.

Fundamentally, a change from discrete-time to continuous-time is not only an implementation change, but also an algorithmic change. Figure 3c shows a continuous-time state decoder whose observed behavior is qualitatively similar to a discrete-time decoder. The delay circuit, labeled **(3)**, uses a linear transconductance amplifier in a follower-integrator configuration. The time constant of this delay circuit should be set to the frame rate of the corresponding discrete-time state decoder.

For correct decoder behavior over the full range of input probability values, the transconductance amplifer in the delay circuit must have a wide differential-input-voltage linear range. In the test chip presented in this paper, an amplifier with a small linear range was used. To work around the problem, we restricted the input probability currents in our experiments to a small multiple of $I_l$.

Figure 4 shows a state decoding system that corresponds to the grammar shown in Figure 1. Each numbered circle corresponds to the circuit shown in Figure 3c. The signal flows of this architecture support a dense layout: a rectangular array of single-state decoding circuits, with input current signal entering from the top edge of the array, and end-state log likelihood outputs exiting from the right edge of the array. States connect to their neighbors via the $V_{i-1}(t)$ and $V_i(t)$ signals shown in Figure 3c. For notational convenience, in this figure we define the unidirectional current $p_i(t)$ to be $I_h b_i(t)$.

In addition to the single-state decoder circuit, several other circuits are required. The "Recurrent Connection" block in Figure 4 implements the loopback connecting the filled circles in Figure 1. We implement this block using a 3-input version of the voltage follower circuit labeled **(1)** in Figure 3c. A simple arithmetic circuit implements the "Word Detect" block. To complete the system, a high fan-in/fan-out control circuit implements the renormalization algorithm. The circuit takes as input the log likelihood signals from all states in the system, and returns the binary signal $V$ to the control input of all states. This control signal determines whether the single-state decoding circuits exhibit normal behavior (Equation 3a) or renormalization behavior (Equation 3b).

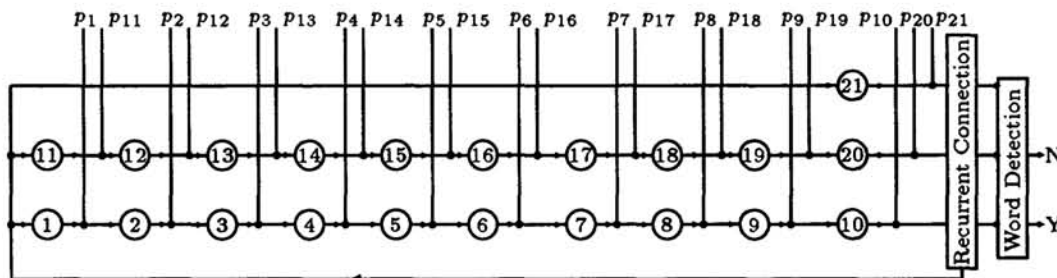

**Figure 4.** State decoder system for grammar shown in Figure 1.

## 3. STATE DECODER TEST CHIP

We fabricated a state decoder test chip in the $2\mu$m, n-well process of Orbit Semiconductor, via MOSIS. The chip has been fully tested and is functional. The chip decodes a grammar consisting of eight ten-state word models and a filler state. The state decoding and word detection sections of the chip contain 2000 transistors, and measure $586 \times 2807\mu$m ($586 \times 2807\lambda$, $\lambda = 1.0\mu$m). In this section, we show test results from the chip, in which we apply a temporal pattern of probability currents to the ten states of one word in the model (numbered 1 through 10) and observe the log likelihood voltage of the final state of the word (state 10).

Figure 5 contains simulated results, allowing us to show internal signals in the system. Figure 5a shows the temporal pattern of input probability currents $p_1 \ldots p_{10}$ that correspond to a simulated input word. Figure 5b shows the log likelihood voltage waveform for the end-state of the word (state 10). The waveform plateaus at $L_h$, the limit of the operating range of the state decoder system. During this plateau this state has the largest log likelihood in the system. Figure 5c is an expanded version of Figure 5b, showing in detail the renormalization cycles. Figure 5d shows the output computed by the "Word Detect" block in Figure 4. Note the smoothness of the waveform, unlike Figure 5c. By subtracting the filler-state log likelihood from the end-state log likelihood, the Word Detect block cancels the common-mode renormalization waveform.

Figure 6 shows a series of four experiments that confirm the qualitative behavior of the state decoder system. This figure shows experimental data recorded from the fabricated test chip. Each experiment consists of playing a particular pattern of input probability currents $p_1 \ldots p_{10}$ to the state decoder many times; for each repetition, a certain aspect of the playback is systematically varied. We measure the peak value of the end state log likelihood during each repetition, and plot this value as a function of the varied input parameter. For each experiment shown in Figure 6, the left plot describes the input pattern, while the right plot is the measured end-state log likelihood data. The experiment shown in Figure 6a involves presenting complete word patterns of varying durations to the decoder. As expected, words with unrealistically short durations have end-state responses below $L_h$, and would not produce successful word detection.

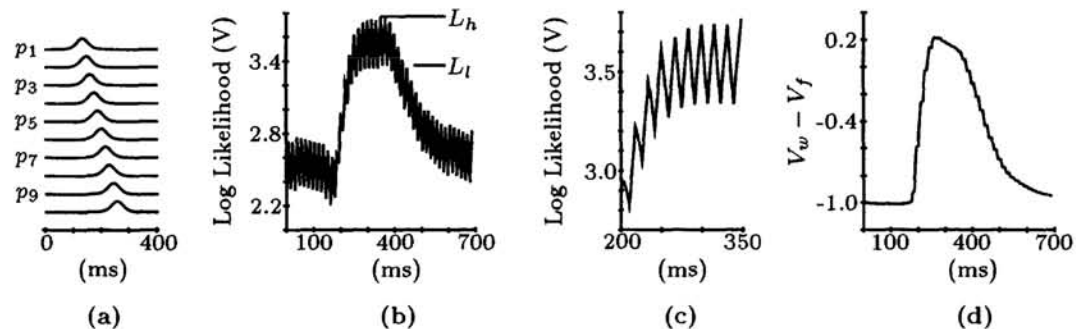

**Figure 5.** Simulation of state decoder: **(a)** Inputs patterns, **(b)**, **(c)** End-state response, **(d)** Word-detection response.

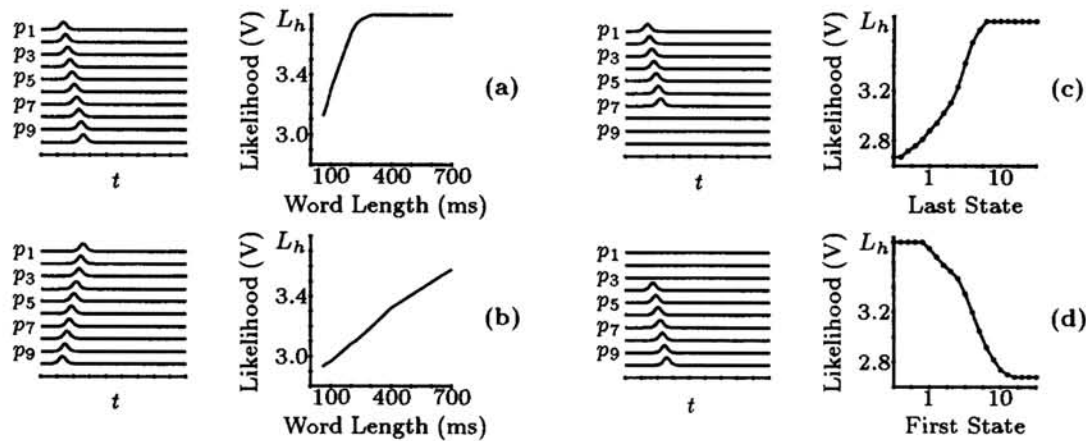

**Figure 6.** Measured chip data for end-state likelihoods for long, short, and incomplete pattern sequences.

The experiment shown in Figure 6b also involves presenting patterns of varying durations to the decoder, but the word patterns are presented "backwards," with input current $p_{10}$ peaking first, and input current $p_1$ peaking last. The end-state response never reaches $L_h$, even at long word durations, and (correctly) would not trigger a word detection.

The experiments shown in Figure 6c and 6d involve presenting partially complete word patterns to the decoder. In both experiments, the duration of the complete word pattern is 250 ms. Figure 6c shows words with truncated endings, while Figure 6d shows words with truncated beginnings. In Figure 6c, end-state log likelihood is plotted as a function of the last excited state in the pattern; in Figure 6d, end-state log likelihood is plotted as a function of the first excited state in the pattern. In both plots the end-state log likelihood falls below $L_h$ as significant information is removed from the word pattern.

While performing the experiments shown in Figure 6, the state-decoder and word-detection sections of the chip had a measured average power consumption of 141 nW ($V_{dd} = 5V$). More generally, however, the power consumption, input probability range, and the number of states are related parameters in the state decoder system.

## Acknowledgments

We thank Herve Bourlard, Dan Hammerstrom, Brian Kingsbury, Alan Kramer, Nelson Morgan, Stylianos Perissakis, Su-lin Wu, and the anonymous reviewers for comments on this work. Sponsored by the Office of Naval Research (URI-N00014-92-J-1672) and the Department of Defense Advanced Research Projects Agency. Opinions, interpretations, conclusions, and recommendations are those of the authors and are not necessarily endorsed by the United States Air Force.

## Reference

Lippmann, R. P., Chang, E. I., and Jankowski, C. R. (1994). "Wordspotter training using figure-of-merit back-propagation," *Proceedings International Conference on Acoustics, Speech, and Signal Processing*, Vol. 1, pp. 389-392.